# Copula Bayesian Networks

**Gal Elidan**
Department of Statistics
Hebrew University
Jerusalem, 91905, Israel
galel@huji.ac.il

## Abstract

We present the Copula Bayesian Network model for representing multivariate continuous distributions, while taking advantage of the relative ease of estimating univariate distributions. Using a novel copula-based reparameterization of a conditional density, joined with a graph that encodes independencies, our model offers great flexibility in modeling high-dimensional densities, while maintaining control over the form of the univariate marginals. We demonstrate the advantage of our framework for generalization over standard Bayesian networks as well as tree structured copula models for varied real-life domains that are of substantially higher dimension than those typically considered in the copula literature.

## 1 Introduction

Multivariate real-valued distributions are of paramount importance in a variety of fields ranging from computational biology and neuro-science to economics to climatology. Choosing and estimating a useful form for the marginal distribution of each variable in the domain is often a straightforward task. In contrast, aside from the normal representation, few univariate distributions have a convenient multivariate generalization. Indeed, modeling and estimation of flexible (skewed, multi-modal, heavy tailed) high-dimensional distributions is still a formidable challenge.

Copulas [23] offer a general framework for constructing multivariate distributions using *any* given (or estimated) univariate marginals and a copula function $C$ that links these marginals. The importance of copulas is rooted in Sklar's theorem [29] that states that any multivariate distribution can be represented as a copula function of its marginals. The constructive converse is important from a modeling perspective as it allows us to *separate* the choice of the marginals and that of the dependence structure which is expressed in $C$. We can, for example, robustly estimate marginals using a non-parametric approach, and then use only few parameters to capture the dependence structure. This can result in a model that is easier to estimate and less prone to over-fitting than a fully non-parametric one, while at the same time avoiding the limitations of a fully parameterized distribution. In practice, copula constructions often lead to significant improvement in density estimation. Accordingly, there has been a dramatic growth of academic and practical interest in copulas in recent years, with applications ranging from mainstream financial risk assessment and actuarial analysis (e.g., Embrechts et al. [7]) to off-shore engineering (e.g., Accioly and Chiyoshi [2]).

Despite the generality of the framework, constructing high-dimensional copulas is difficult, and much of the research involves only the bivariate case. Several works have attempted to overcome this difficulty by suggesting innovative ways in which bivariate copulas can be combined to form workable copulas of higher dimensions. These attempts, however, are either limited to hierarchical [26] or mixture of trees [14] compositions, or rely on a recursive construction of conditional bivariate copulas [1, 3, 17] that is somewhat elaborate for high dimensions. In practice, applications are almost always limited to a modest ($< 10$) number of variables (see Section 6 for further discussion).

Bayesian networks (BNs) [25] offer a markedly different approach for representing multivariate distributions. In this widely used framework, a graph structure encodes independencies which imply a decomposition of the joint density into local terms (the density of each variable conditioned on its

parents). This decomposition in turn facilitates efficient probabilistic computation and estimation, making the framework amenable to high-dimensional domains. However, the expressiveness of these models is hampered by practical considerations that almost always lead to the the reliance on simple parametric forms. Specifically, non-parametric variants of BNs (e.g., [9, 27]) typically involve elaborate training setups with a running time that grows unfavorably with the number of samples and local graph connectivity. Furthermore, aside from the case of the normal distribution, the form of the univariate marginal is neither under control nor is it typically known.

Our goal is to construct flexible multivariate continuous distributions that maintain desired marginals while accommodating tens and hundreds of variables, or more. We present Copula Bayesian Networks (CBNs), an elegant marriage between the copula and the Bayesian network frameworks.[1] As in BNs, we make use of a graph to encode independencies that are assumed to hold. Differently, we rely on local copula functions and an explicit globally shared parameterization of the univariate densities. This allows us to retain the flexibility of BNs, while offering control over the form of the marginals, resulting in substantially improved multivariate densities (see Section 7 for a discussion of the related works of Kirshner [14] and Liu et al. [20]).

At the heart of our approach is a novel reparameterization of a conditional density using a copula quotient. With this construction, we prove a parallel to the BN factorization theorem: a decomposition of the joint density according to the structure of the graph implies a decomposition of the joint copula. Conversely, a product of local copula-based quotient terms is a valid multivariate copula. This result provides us with a flexible modeling tool where joint densities are constructed via a composition of local copulas and marginal densities. Importantly, the construction also allows us to use standard BN machinery for estimation and structure learning. Thus, our model opens the door for flexible explorative learning of high-dimensional models that retain desired marginal characteristics.

We learn the structure and parameters of a CBN for three varied real-life domains that are of a significantly higher dimension than typically reported in the copula literature. Using standard copula functions, we show that in all cases our approach leads to consistent and significant improvement in generalization when compared to standard BN models as well as a tree-structured copula model.

## 2   Copulas

Let $\mathcal{X} = \{X_1, \ldots, X_N\}$ be a finite set of real-valued random variables and let $F_{\mathcal{X}}(\mathbf{x}) \equiv P(X_1 \leq x_1, \ldots, X_n \leq x_N)$ be a (cumulative) distribution function over $\mathcal{X}$, with lower case letters denoting assignment to variables. By slight abuse of notation, we use $F(x_i) \equiv F(X_i \leq x_i, X_{\mathcal{X}/X_i} = \infty)$ and $f(x_i) \equiv f_{X_i}(x_i)$, and similarly for sets of variables $f(\mathbf{y}) \equiv f_{\mathbf{Y}}(\mathbf{y})$. A copula function [23, 29] links marginal distributions to form a multivariate one. Formally,

**Definition 2.1:** Let $U_1, \ldots, U_N$ be real random variables marginally uniformly distributed on $[0, 1]$. A copula function $C : [0, 1]^N \rightarrow [0, 1]$ is a joint distribution function

$$C(u_1, \ldots, u_N) = P(U_1 \leq u_1, \ldots, U_N \leq u_N)$$ ∎

Copulas are important because of the following seminal result

**Theorem 2.2:   [Sklar 1959]** *Let $F(x_1, \ldots, x_N)$ be any multivariate distribution over real-valued random variables, then there exists a copula function such that*

$$F(x_1, \ldots, x_N) = C(F(x_1), \ldots, F(x_N)).$$

*Furthermore, if each $F(x_i)$ is continuous then $C$ is unique.*

The constructive converse which is of central interest from a modeling perspective is also true: since for *any* random variable the cumulative distribution $F(x_i)$ is uniformly distributed on $[0, 1]$, *any* copula function taking the marginal distributions $\{F(x_i)\}$ as its arguments, defines a valid joint distribution with marginals $F(x_i)$. Thus, copulas are "distribution-generating" functions that allow us to separate the choice of the univariate marginals and that of the dependence structure expressed in the copula function $C$, often resulting in an effective real-valued construction.[2].

Figure 1: Samples from the 2-dimensional normal copula density using a correlation matrix with a unit diagonal and an off-diagonal coefficient of 0.25. (left) with zero mean and unit variance normal marginals; (right) with a mixture of two Gaussians marginals.

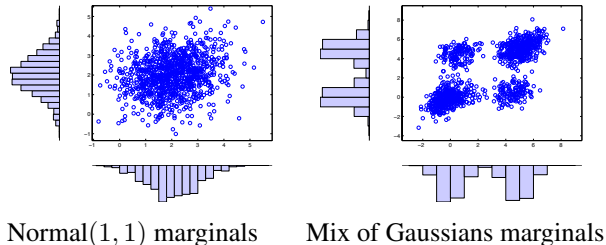

Normal$(1,1)$ marginals     Mix of Gaussians marginals

To derive the joint *density* $f(\mathbf{x}) = \frac{\partial^N F(\mathbf{x})}{\partial x_1 \dots \partial x_N}$ from the copula construction, assuming $F$ has N-order partial derivatives (true almost everywhere when $F$ is continuous), and using the chain rule, we have

$$f(\mathbf{x}) = \frac{\partial^N C(F(x_1), \dots, F(x_N))}{\partial F(x_1) \dots \partial F(x_N)} \prod_i f(x_i) = c(F(x_1), \dots, F(x_N)) \prod_i f(x_i), \qquad (1)$$

where $c(F(x_1), \dots, F(x_N))$, is called the *copula density function*. Eq. (1) will be of central use in this paper as we will directly model joint densities.

**Example 2.3:** A simple copula widely explored in the financial community is the Gaussian copula constructed directly by inverting Sklar's theorem [7]

$$C(\{F(x_i)\}) = \Phi_\Sigma \left( \Phi^{-1}(F(x_1)), \dots, \Phi^{-1}(F(x_N)) \right), \qquad (2)$$

where $\Phi$ is the standard normal distribution and $\Phi_\Sigma$ is the zero mean normal distribution with correlation matrix $\Sigma$. To get a sense of the power of copulas, Figure 1 shows samples generated from this copula using two different families of univariate marginals. More generally and without added computational difficulty, we can also mix and match marginals of different forms.

## 3 Copula Bayesian Networks (CBNs)

As in the copula framework, our goal is to model real-valued multivariate distributions while taking advantage of the relative ease of one dimensional estimation. To cope with high-dimensional domains, as in BNs, we would also like to utilize independence assumptions encoded by a graph. To achieve this goal, we will construct multivariate copulas that are a composition of local copulas that follow the structure of the graph. We start with the building block of our construction.

### 3.1 Copula Parameterization of The Conditional Density

As in the BN framework, the building block of our model will be a local conditional density. We start with a parameterization of such a density using copulas:

**Lemma 3.1:** *Let $f(x \mid \mathbf{y})$, with $\mathbf{y} = \{y_1, \dots, y_K\}$, be a conditional density function and let $f(x)$ be the marginal density of $X$. Then there exists a copula density function $c(F(x), F(y_1), \dots, F(y_K))$ such that*

$$f(x \mid \mathbf{y}) = R_c(F(x), F(y_1), \dots, F(y_K)) f(x)$$

*where $R_c$ is the ratio*

$$R_c(F(x), F(y_1), \dots, F(y_K)) \equiv \frac{c(F(x), F(y_1), \dots, F(y_K))}{\int c(F(x), F(y_1), \dots, F(y_K)) f(x) dx} = \frac{c(F(x), F(y_1), \dots, F(y_K))}{\frac{\partial^K C(1, F(y_1), \dots, F(y_K))}{\partial F(y_1) \dots \partial F(y_K)}},$$

*and where $R_c$ is defined to be 1 when $\mathbf{y} = \emptyset$. The converse is also true, for any copula density function $c$, $R_c(F(x), F(y_1), \dots, F(y_K)) f(x)$ defines a valid conditional density function.*

Before proving this result, it is important to understand why the derivative form of denominator (right-most term) is more useful than the standard normalization integral $\int c(F(x), F(y_1), \dots, F(y_K)) f(x) dx$. Recall that $c()$ is itself an $N$-order derivative of the copula function so computing our denominator is no more difficult than computing $c()$. Indeed, for the majority of existing copula functions, both have an explicit form. In contrast, the integral term depends both on the copula form and the univariate marginal, and is generally difficult to compute.

**Proof:** From the basic properties of cumulative distribution functions, we have that for *any* copula function $C(1, F(y_1), \ldots, F(y_K)) = F(y_1, \ldots, y_k)$ and thus, using the derivative chain rule,

$$f(\mathbf{y}) = \frac{\partial^K C(1, F(y_1), \ldots, F(y_K))}{\partial y_1, \ldots, y_K} = \frac{\partial^K C(1, F(y_1), \ldots, F(y_K))}{\partial F(y_1) \ldots \partial F(y_K)} \prod_k f(y_k).$$

From Eq. (1) we have that there exists a copula density for which $f(x, y_1, \ldots, y_K) = c(F(x), F(y_1), \ldots, F(y_K))f(x) \prod_k f(y_k)$. It follows that there exists a copula for which

$$
\begin{aligned}
f(x \mid \mathbf{y}) &= \frac{f(x, y_1, \ldots, y_K)}{f(\mathbf{y})} = \frac{c(F(x), F(y_1), \ldots, F(y_K))f(x) \prod_k f(y_k)}{\frac{\partial^K C(1, F(y_1), \ldots, F(y_K))}{\partial F(y_1) \ldots \partial F(y_K)} \prod_k f(y_k)} \\
&= \frac{c(F(x), F(y_1), \ldots, F(y_K))f(x)}{\frac{\partial^K C(1, F(y_1), \ldots, F(y_K))}{\partial F(y_1) \ldots \partial F(y_K)}} \equiv R_c(F(x), F(y_1), \ldots, F(y_K))f(x)
\end{aligned}
$$

As in Sklar's theorem and $Eq.$ (1), the converse follows easily by reversing the arguments. ∎

The implications of this result will underlie our construction: *any* copula density function $c(x, y_1, \ldots, y_K)$, together with $f(x)$, can be used to parameterize a conditional density $f(x \mid \mathbf{y})$.

### 3.2 Decomposition of The Joint Copula

Let $\mathcal{G}$ be a directed acyclic graph whose nodes correspond to the random variables $\mathcal{X}$, and let $\mathbf{Pa}_i = \{\mathbf{Pa}_{i1}, \ldots, \mathbf{Pa}_{ik_i}\}$ be the parents of $X_i$ in $\mathcal{G}$. $\mathcal{G}$ encodes the independence statements $I(\mathcal{G}) = \{(X_i \perp NonDescendants_i \mid \mathbf{Pa}_i)\}$, where $NonDescendants_i$ are nodes that are non-descendants of $X_i$ in $\mathcal{G}$. We say that $f_{\mathcal{X}}(\mathbf{x})$ decomposes according to $\mathcal{G}$ if it can be written as a product of conditional densities $f_{\mathcal{X}}(\mathbf{x}) = \prod_i f(X_i \mid \mathbf{Pa}_i)$. It can be shown that if $f$ decomposes according to $\mathcal{G}$ then $I(\mathcal{G})$ hold in $f_{\mathcal{X}}(\mathbf{x})$. The converse is also true: if $I(\mathcal{G})$ hold in $f_{\mathcal{X}}(\mathbf{x})$ then the density decomposes according to $\mathcal{G}$ (see [16], theorems 3.1 and 3.2). These results form the basis for the BN model [25] where a joint density is constructed via a composition of local conditional densities. We now show that similar results hold for a multivariate copula. This in turn will provide the basis for our construction of the CBN model.

**Theorem 3.2 :** **Decomposition**. *Let $\mathcal{G}$ be a directed acyclic graph over $\mathcal{X}$, and let $f_{\mathcal{X}}(\mathbf{x})$ be parameterized via a joint copula density $f_{\mathcal{X}}(\mathbf{x}) = c(F(x_1), \ldots, F(x_N)) \prod_i f(x_i)$, with $f_{\mathcal{X}}(\mathbf{x})$ strictly positive for all values of $\mathcal{X}$. If $f_{\mathcal{X}}(\mathbf{x})$ decomposes according to $\mathcal{G}$ then the copula density $c(F(x_1), \ldots, F(x_N))$ also decomposes according to $\mathcal{G}$*

$$c(F(x_1), \ldots, F(x_N)) = \prod_i R_{c_i}(F(x_i), \{F(\mathbf{pa}_{ik})\}),$$

*where $c_i$ is a local copula that depends only on the value of $X_i$ and its parents in $\mathcal{G}$.*

**Proof:** Using the positivity assumption, we can rearrange Eq. (1) to get $c(F(x_1), \ldots, F(x_N)) = \frac{f(\mathbf{x})}{\prod_i f(x_i)}$. From Lemma 3.1 and the decomposition of $f(\mathbf{x})$ we have

$$
\begin{aligned}
c(F(x_1), \ldots, F(x_N)) &= \frac{f(\mathbf{x})}{\prod_i f(x_i)} = \frac{\prod_i f(x_i \mid \mathbf{pa}_i)}{\prod_i f(x_i)} \\
&= \frac{\prod_i R_{c_i}(F(x_i), \{F(\mathbf{pa}_{ik})\})f(x_i)}{\prod_i f(x_i)} = \prod_i R_{c_i}(F(x_i), \{F(\mathbf{pa}_{ik})\})
\end{aligned}
$$

∎

The constructive converse that is of central interest here is also true:

**Theorem 3.3 :** **Composition**. *Let $\mathcal{G}$ be a directed acyclic graph over $\mathcal{X}$. In addition, let $\{c_i(F(x_i), F(\mathbf{pa}_{i1}), \ldots, F(\mathbf{pa}_{ik_i}))\}$ be a set of strictly positive copula densities associated with the nodes of $\mathcal{G}$ that have at least one parent. If $I(\mathcal{G})$ hold then the function*

$$g(F(x_1), \ldots, F(x_N)) = \prod_i R_{c_i}(F(x_i), \{F(\mathbf{pa}_{ik})\}),$$

*is a valid copula density $c(F(x_1), \ldots, F(x_N))$ over $\mathcal{X}$.*

This above theorem can be proved directly via induction or using our reparameterization lemma and standard BN results. It is important to note that the local copulas *do not* need to agree on the non-univariate marginals of overlapping variables. This is a result of the fact that each copula $c_i$ only appears as part of a quotient term which is used to parameterize a *conditional* density. This gives us the freedom to mix and match local copulas of different types. Equally important is the fact that aside from the univariate densities, we do not need to concern ourselves with any marginal constraints when estimating the parameters of these local copulas functions.

### 3.3 A Multivariate Copula Model

We are now ready to construct a joint density given univariate marginals by properly composing local terms and without worrying about global coherence:

**Definition 3.4:** A Copula Bayesian Network (CBN) is a triplet $\mathcal{C} = (\mathcal{G}, \Theta_C, \Theta_f)$ that encodes the joint density $f_{\mathcal{X}}(\mathbf{x})$. $\Theta_C$ is a set of local copula densities functions $c_i(F(x_i), \{F(\mathbf{pa}_{ik})\})$ that are associated with the nodes of $\mathcal{G}$ that have at least one parent. $\Theta_f$ is the set of parameters representing the marginal densities $f(x_i)$. $f_{\mathcal{X}}(\mathbf{x})$ is parameterized as

$$f_{\mathcal{X}}(\mathbf{x}) = \prod_i R_{c_i}(F(x_i), \{F(\mathbf{pa}_{ik})\})f(x_i). \qquad \blacksquare$$

Using our previous developments and applying Eq. (1) to $f_{\mathcal{X}}(\mathbf{x})$, we have:

**Corollary 3.5:** *A Copula Bayesian Network defines a valid joint density $f_{\mathcal{X}}(\mathbf{x})$ whose marginal distributions are parameterized by $\Theta_f$ and where the independence statements $I(\mathcal{G})$ hold.*

The main difference between the CBN model and a regular BN, aside from a novel choice for the local conditional parameterization, is in the shared *global* component that has the explicit semantics of the univariate marginals. Concretely, the CBN model allows us to decompose the problem of representing a multivariate distribution with *given* (or estimated) univariate marginals into many local problems that, depending on the structure of $\mathcal{G}$, can be substantially smaller in dimension.

For each family of $X_i$ and its parents we are still faced with the problem of choosing an appropriate local copula. In this work we simply limit ourselves to copulas that have convenient multivariate form, but any of the recently suggested methods for constructing multivariate copulas functions (see Section 6) can also be used. In either case, limiting ourselves to a smaller number of variables (a node and its parents) makes the construction of the local copula substantially easier than the construction of the full copula over $\mathcal{X}$. Importantly, as in the case of BNs, our construction of a joint copula density that decomposes over the graph structure $\mathcal{G}$ also facilitates efficient parameter estimation and model selection (structure learning), as we briefly discuss in the next section.

## 4 Learning

As in the case of BNs, the product form of our CBN facilitates relatively efficient estimation and model selection. The machinery is standard and only briefly described below.

**Parameter Estimation**
Given a complete dataset $\mathcal{D}$ of $M$ instances where all of the variables $\mathcal{X}$ are observed in each instance, the log-likelihood of the data given a CBN model $\mathcal{C}$ is

$$\ell(\mathcal{D} : \mathcal{C}) = \sum_{m=1}^M \sum_i \log f(x_i[m]) + \sum_{m=1}^M \sum_i \log R_i(F(x_i)[m], F(\mathbf{pa}_{i1}[m]), \dots, F(\mathbf{pa}_{ik_i}[m]))$$

While this objective appears to fully decompose according to the structure of $\mathcal{G}$, each marginal distribution $F(x_i)$ actually appears in several local copula terms (of $X_i$ and its children in $\mathcal{G}$). To facilitate efficient estimation, we adopt the common approach where the marginals are estimated first [13]. Given $F(x_i)$, we can then estimate the parameters of each local copula *independently* of the others. We estimate the univariate densities using a standard normal kernel-based approach [24].

In this work we consider two of the simplest and most commonly used copula functions. For Frank's Archimedean copula $C(u_1, \dots, u_N) = -\frac{1}{\theta} \log \left(1 + \prod_i (e^{-\theta F(x_i)} - 1)/(e^{-\theta} - 1)^{N-1}\right)$, and for the Gaussian copula (see Section 2) with a uniform correlation parameter, we find the maximum

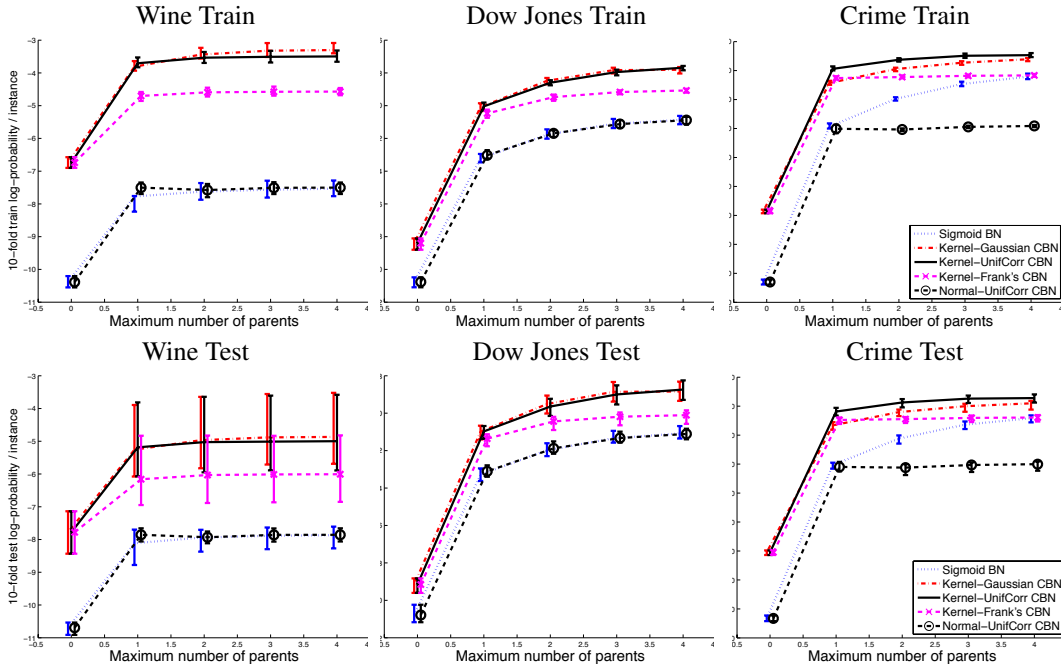

Figure 2: Train and test set performance for the 12 variable **Wine**, 28 variable **Dow Jones** and 100 variables **Crime** datasets. Models compared: Sigmoid BN; CBN with a uniform correlation normal copula (single parameter); CBN with a full normal copula ($0.5 * d(d-1)$ parameters); CBN with Frank's single parameter copula. Shown is the 10-fold average log-probability per instance (y-axis) vs. the maximal number of parents allowed in the network (x-axis). Error bars (slightly shifted for readability) show the $10 - 90\%$ range. The structure for all models was learned with the same search procedure using the BIC model selection score.

likelihood parameters using a standard conjugate gradient algorithm. For the Gaussian copula with a full covariance matrix, a reasonably effective and substantially more efficient method is based on the relationship between the copula function and Kendall's Tau dependence measure [19]. For lack of space, further details for both of these copulas are provided in the supplementary material.

**Model Selection**

Very briefly, to learn the structure of $\mathcal{G}$, we use a standard score-based approach that starts with the empty network, and greedily advances via local modifications to the current structure (add/delete/reverse edge). The search is guided by the Bayesian information criterion [28] that balances the likelihood of the model and its complexity $\text{score}(\mathcal{G} : \mathcal{D}) = \ell(\mathcal{D} : \hat{\theta}, \mathcal{G}) - \frac{1}{2}\log(M)|\Theta_{\mathcal{G}}|$, where $\hat{\theta}$ are the maximum-likelihood parameters, and $|\Theta_{\mathcal{G}}|$ is the number of free parameters associated with the graph structure $\mathcal{G}$. During the search, we also use a TABU list and random restarts [10] to mitigate the problem of local maxima. See Koller and Friedman [16] for more details.

## 5 Experimental Evaluation

We assess the effectiveness of our approach for density estimation by comparing CBNs and BNs learned from training data in terms of log-probability performance on test data. For BNs, we use a linear Gaussian conditional density and a non-linear Sigmoid one (see Koller and Friedman [16]). For CBNs, to demonstrate the flexibility of our framework, we consider the three local copula functions discussed in Section 4: fully parametrized **Normal** copula; the same copula with a single correlation parameter and unit diagonal (**UnifCorr**); **Frank's** single parameter Archimedean copula. We use standard normal kernel density estimation for the univariate densities. The structure of both the BN and CBN models was learned using the same greedy structure search procedure described in Section 4. We consider three datasets of a markedly different nature and dimensionality:

- **Wine Quality** (UCI repository). 11 physiochemical properties and a sensory quality variable for the red Portuguese "Vinho Verde" wine [4]. Included are measurements from 1599 tastings.

Figure 3: Comparison of the number of edges learned in the different random run for different models (y-axis) vs. the Sigmoid BN model (x-axis), when the maximal number of parents in the network was limited to 4.

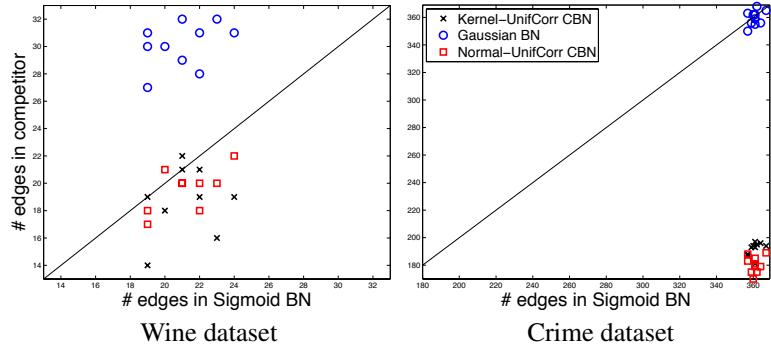

Wine dataset                    Crime dataset

- **Dow Jones**. 2001-2005 (1508 trading days) daily adjusted changes of the 30 index stocks. To avoid arbitrary imputation, two stocks not traded in all of these days were excluded (KFT,TRV).
- **Crime** (UCI repository). 100 observed variables relating to crime ranging from household size to fraction of children born outside of a marriage, for 1994 communities across the U.S.

Figure 2 compares average log-probability (y-axis) for 10 random equal train/test splits as a function of the maximal number of parents allowed in the network (x-axis). Results for the linear Gaussian BN were almost identical to those of the sigmoid BN for the **Wine** and **Dow Jones** datasets and inferior for the **Crime** dataset, and are omitted for clarity. For all datasets, the copula based models offer a clear gain in training performance as well as in generalization on unseen test instances. Remarkably, the single parameter (for each local density) **UnifCorr** model is superior to the BN model even when the latter utilizes up to 8 local parameters (with 4 parents). In fact, even Frank's single parameter Archimedean copula which is constrained by the fact that all of its $K$-marginals are equal [23], is superior to the BN model. Importantly, the advantage of the CBN model is significant as the units of improvement are in bits/instance. That is, an improvement of 2 bits/instance translates into *each* test instance being, on average, four times as likely.[3] It is also important to note the benefit that comes with structures that are richer than a tree. As the number of allowed parents (x-axis) is increased, gains are relatively small when the dimensionality of the domain is limited (12 variables); The gains are, however, quite substantial for the more complex domains.

To understand the role of the univariate marginals, we start with the no dependency network (0 on x-axis), where the advantage of CBNs is solely due to the use of flexible univariate marginals. Surprisingly, even with single parameter copulas, although much simpler than the Sigmoid form used for the BN model, we are able to maintain much of that advantage as the model becomes more complex. As expected, this is not the case when we constrain the CBN model to have normal marginals (**Normal-UnifCorr**) and when the domain is sufficiently complex (**Crime**).

To get a sense of the overall dependency structure, Figure 3 shows the number of edges learned for the different models. For the **Wine** dataset, the linear BN attempts to compensate for its constrained form by using substantially more edges than the non-linear Sigmoid BN. The **Kernel-UnifCorr** CBN, in contrast, tends to use less edges while achieving higher test performance. Finally, the **Normal-UnifCorr** CBN model, despite the *forced* normal marginals, does not lead to overly complex structures as it is constrained by the simplicity of the copula function (single parameter). For the challenging **Crime** dataset, the differences are more pronounced: both the linear and non-linear BN models almost saturate the limit of 4 parents per variable, while the **Kernel-UnifCorr** copula model requires, on average, less than half the number of parents to achieve superior performance.

Finally, in Figure 4, we demonstrate the qualitative advantage of CBNs by comparing empirical values from the *test* data (left) with samples generated from the different models. For the 'physical density' and 'alcohol' variables (top), the CBN samples (middle) are better than the BN ones (right), but not dramatically so. However, for the 'residual sugar' and 'physical density' pair (bottom), where the empirical dependence is far from normal, the advantage of the CBN representation is clear. We recall that the CBN model uses a simple normal copula so that the advantage is solely rooted in the distortion of the input to the copula created by the kernel-based univariate representation. With more expressive copulas we can expect further qualitative and quantitative advantages.

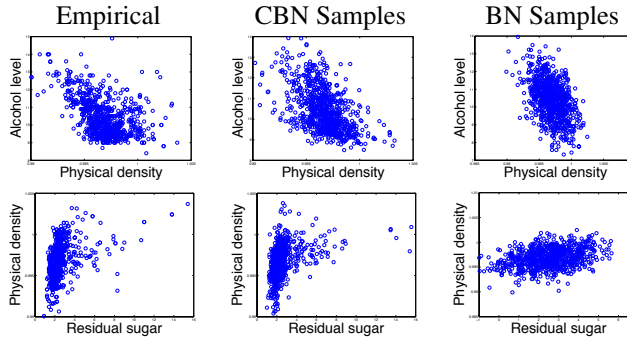

Figure 4: Demonstration of the dependency learned for the **Wine** dataset for two variable pairs. Compared is the empirical distribution in the **test** data (left) with samples generated from the learned CBN (middle) and BN (right) models. To eliminate the effect of differences in structure, the CBN model was forced to use the structure learned for the BN model which contains the network fragment 'residual sugar' → 'physical density' → 'alcohol level'.

## 6   Related Work

For lack of space we do not discuss direct multivariate copula constructions (e.g., [8, 15, 18, 22]) that are typically effective only for few dimensions, and focus on composite constructions that build on smaller (bivariate) copulas. The Vine model [3] relies on a recursive construction of bivariate copulas to parameterize a multivariate one. Although it uses a graphical representation, the framework is inherently different from ours: *conditional independence* is replaced with a *conditional dependence* whose parameters depend on the conditioning variable(s). Kurwicka and Cooke [17] reveal a direct connection between vines and belief networks, but that is limited to the scenario of elliptical bivariate copulas. Relying on the same representation, Aas et al. [1] suggest an alternative construction methodology. While the vine representation is certainly general, the need to condition on many variables using a somewhat elaborate construction limits practical applications to a modest number of variables. Aas et al. [1] do note the simplification that can result from making independence assumptions, but do not provide a general framework for doing so. Savu and Trede [26] suggest an alternative model that is limited to a hierarchical tree structure of bivariate Archimedean copulas.

Kirshner [14] uses the copula product operator of Darsow et al. [5] to suggest a mixture of trees model that is directly motivated by the field of graphical models. The relationship between our model to theirs is the same as that of a general BN to a mixture of trees model [21]. Most recently, Liu et al. [20] consider a general sparse undirected copula-based model that is focused on the semi and non-parametric aspect of modeling, and is specific to the case of the normal copula.

Finally, it is important to put the dimension of the domains we consider in this work (up to 100 variables) in perspective. Copula applications are numerous yet most are limited to a relatively small number ($< 10$) of variables. Heinen and Alfonso [11] are unique in that they consider 95 variables, but using an approach that is tailored to the specific details of the GARCH model.

## 7   Discussion and Future Work

We presented Copula Bayesian Networks, a marriage between the Bayesian network and copula frameworks. Building on a novel reparameterization of the conditional density, our model offers great flexibility in modeling high-dimensional continuous distribution while offering control over the form of the univariate marginals. We applied our approach to three markedly different real-life datasets and, in all cases, demonstrated a consistent and significant generalization advantage.

Our contribution is threefold. First, our framework allows us to flexibly "mix and match" local copulas and univariate densities of any form. Second, like BNs, we allow for independence assumptions that are more expressive than those possible with tree-based constructions, leading to generalization advantages. Third, we leverage on existing machinery to perform model selection in significantly higher dimensions than typically considered in the copula literature. Thus, our work opens the door for numerous applications where the flexibility of copulas is needed but could not be previously utilized. In a companion paper [6], we also show that CBNs give rise to an efficient inference procedure.

The gap between train and test performance for CBNs motivates the development of model selection scores tailored to the copula framework (e.g., based on rank correlation). It would also be interesting to see if our framework can be adapted to the cumulative scenario, while allowing for independencies quite different from the recently introduced cumulative network model [12].

## Acknowledgements

I am grateful to Ariel Jaimovich, Amir Globerson, Nir Friedman and Fabio Spizzichino for their comments on earlier drafts of this manuscript. G. Elidan was supported by the Alon fellowship.

## Footnotes

[1] A preliminary draft of this paper appeared as a technical report. A companion paper [6] addresses the question of performing approximate inference in Copula Bayesian networks.

[2] Copulas can also be defined given non-continuous marginals and for ordinal random variables. These extensions are orthogonal to our work and to maintain clarity we focus here on the continuous case

[3]Note that the performance for the crime domain is on an unusually high scale since some of the variables are closely correlated, leading to peaked densities. We emphasize that this does not effect the *relative* merit of a method - an advantage of a bit/instance still translates to each instance being, on average, twice as likely.

## References

[1] K. Aas, C. Czado, A. Frigessi, and H. Bakken. Pair-copula constructions of multiple dependencies. *Insurance: Mathematics and Economics*, 44:182–198, 2009.

[2] R. Accioly and F. Chiyoshi. Modeling dependence with copulas: a useful tool for field development decision process. *Journal of Petroleum Science and Engineering*, 44:83–91, 2004.

[3] T. Bedford and R. Cooke. Vines - a new graphical model for dependent random variables. *Annals of Statistics*, 30(4):1031–1068, 2002.

[4] P. Cortez, A. Cerdeira, F. Almeida, T. Matos, and J. Reis. Modeling wine preferences by data mining from physicochemical properties. *Decision Support Systems*, 47(4):547–553, 2009.

[5] W. Darsow, B. Nguyen, and E. Olsen. Copulas and Markov processes. *Illinois J Math*, 36:600–642, 1992.

[6] G. Elidan. Inference-less density estimation using Copula Bayesian Networks. In *Uncertainty in Artificial Intelligence (UAI)*, 2010.

[7] P. Embrechts, F. Lindskog, and A. McNeil. Modeling dependence with copulas and applications to risk management. *Handbook of Heavy Tailed Distributions in Finance*, 2003.

[8] M. Fischer and C. Kock. Constructing and generalizing given multivariate copulas. Technical report, Working paper, University of Erlangen-Nurnberg, Nurnberg, 2007.

[9] N. Friedman and I. Nachman. Gaussian Process Networks. In *Uncertainty in AI (UAI)*, 2000.

[10] F. Glover and M. Laguna. Tabu search. In C. Reeves, editor, *Modern Heuristic Techniques for Combinatorial Problems*, Oxford, England, 1993. Blackwell Scientific Publishing.

[11] A. Heinen and A. Alfonso. Asymmetric CAPM dependence for large dimensions: The canonical vine autoregressive copula model. ECORE Discussion Paper, 2008.

[12] J. Huang and B. Frey. Cumulative distribution networks and the derivative-sum-product algorithm. In *Uncertainty in Artificial Intelligence (UAI)*, 2008.

[13] H. Joe and J. Xu. The estimation method of inference functions for margins for multivariate models. Technical Report 166, Department of Statistics, University of British Columbia, 1996.

[14] S. Kirshner. Learning with tree-averaged densities and distributions. In *Neural Information Processing Systems (NIPS)*, 2007.

[15] K. Koehler and J. Symanowski. Constructing multivariate distributions with specific marginal distributions. *Journal of Multivariate Distributions*, 55:261–282, 1995.

[16] D. Koller and N. Friedman. *Probabilistic Graphical Models: Principles and Techniques*. MIT, 2009.

[17] D. Kurwicka and R. Cooke. The vine copula method for representing high dimensional dependent distributions: Applications to continuous belief nets. In *The Winter Simulation Conference*, 2002.

[18] E. Liebscher. Modelling and estimation of multivariate copulas. Technical report, Working paper, University of Applied Sciences, Merseburg, 2006.

[19] F. Lindskog, A. McNeil, and U. Schmock. Kendall's tau for elliptical distributions. *Credit Risk - measurement, evaluation and management*, pages 149–156, 2003.

[20] H. Liu, J. Lafferty, and L. Wasserman. The nonparanormal: Semiparametric estimation of high dimensional undirected graphs. *Journal of Machine Learning Research*, 10:22952328, 2010.

[21] M. Meila and M. Jordan. Estimating dependency structure as a hidden variable. In *Neural Information Processing Systems (NIPS)*, 1998.

[22] P. Morillas. A method to obtain new copulas from a given one. *Metrika*, 61:169–184, 2005.

[23] R. Nelsen. *An Introduction to Copulas*. Springer, 2007.

[24] E. Parzen. On estimation of a probability density function and mode. *Annals of Mathematical Statistics*, 33:1065–1076, 1962.

[25] J. Pearl. *Probabilistic Reasoning in Intelligent Systems*. Morgan Kaufmann, 1988.

[26] C. Savu and M. Trede. Hierarchical archimedean copulas. In *the Conf on High Frequency Finance*, 2006.

[27] A. Schwaighofer, M. Dejori, V. Tresp, and M. Stetter. Structure Learning with Nonparametric Decomposable Models. In the *International Conference on Artificial Neural Networks*, 2007.

[28] G. Schwarz. Estimating the dimension of a model. *Annals of Statistics*, 6:461–464, 1978.

[29] A. Sklar. Fonctions de repartition a n dimensions et leurs marges. *Publications de l'Institut de Statistique de L'Universite de Paris*, 8:229–231, 1959.

